# Fast, large-scale transformation-invariant clustering

**Brendan J. Frey**
Machine Learning Group
University of Toronto
www.psi.toronto.edu/∼frey

**Nebojsa Jojic**
Vision Technology Group
Microsoft Research
www.ifp.uiuc.edu/∼jojic

## Abstract

In previous work on "transformed mixtures of Gaussians" and "transformed hidden Markov models", we showed how the EM algorithm in a discrete latent variable model can be used to jointly normalize data (*e.g.*, center images, pitch-normalize spectrograms) and learn a mixture model of the normalized data. The only input to the algorithm is the data, a list of possible transformations, and the number of clusters to find. The main criticism of this work was that the exhaustive computation of the posterior probabilities over transformations would make scaling up to large feature vectors and large sets of transformations intractable. Here, we describe how a tremendous speed-up is acheived through the use of a variational technique for decoupling transformations, and a fast Fourier transform method for computing posterior probabilities. For $N \times N$ images, learning $C$ clusters under $N$ rotations, $N$ scales, $N$ $x$-translations and $N$ $y$-translations takes only $(C + 2 \log N)N^2$ scalar operations per iteration. In contrast, the original algorithm takes $CN^6$ operations to account for these transformations. We give results on learning a 4-component mixture model from a video sequence with frames of size $320 \times 240$. The model accounts for 360 rotations and 76,800 translations. Each iteration of EM takes only 10 seconds per frame in MATLAB, which is over 5 *million* times faster than the original algorithm.

## 1 Introduction

The task of clustering raw data such as video frames and speech spectrograms is often obfuscated by the presence of random, but well-understood transformations in the data. Examples of these transformations include object motion and camera motion in video sequences and pitch modulation in spectrograms.

The machine learning community has proposed a variety of sophisticated techniques for pattern analysis and pattern classification, but these techniques have mostly assumed the data is already normalized (*e.g.*, the patterns are centered in the images) or nearly normalized. Linear approximations to the transformation manifold have

been used to significantly improve the performance of feedforward discriminative classifiers such as nearest neighbors and multilayer perceptrons (Simard, LeCun and Denker 1993). Linear generative models (factor analyzers, mixtures of factor analyzers) have also been modified using linear approximations to the transformation manifold to build in some degree of transformation invariance (Hinton, Dayan and Revow 1997). A multi-resolution approach can be used to extend the usefulness of linear approximations (Vasconcelos and Lippman 1998), but this approach is susceptable to local minima – *e.g.* a pie may be confused for a face at low resolution.

For significant levels of transformation, linear approximations are far from exact and better results can be obtained by explicitly considering transformed versions of the input. This approach has been used to design "convolutional neural networks" that are invariant to translations of parts of the input (LeCun et al. 1998).

In previous work on "transformed mixtures of Gaussians" (Frey and Jojic 2001) and "transformed hidden Markov models" (Jojic et al. 2000), we showed how the EM algorithm in a discrete latent variable model can be used to jointly normalize data (*e.g.*, center video frames, pitch-normalize spectrograms) and learn a mixture model of the normalized data. We found "that the algorithm is reasonably fast (it learns in minutes or hours) and very effective at transformation-invariant density modeling." Those results were for $44 \times 28$ images, but realistic applications such as home video summarization require near-real-time processing of medium-quality video at resolutions near $320 \times 240$. In this paper, we show how a variational technique and a fast Fourier method for computing posterior probabilities can be used to achieve this goal.

## 2  Background

In (Frey and Jojic 2001), we introduced a single discrete variable that enumerates a discrete set of possible transformations that can occur in the input. Here, we break the transformation into a sequence of transformations. $\mathbf{T}_k$ is the random variable for the transformation matrix at step $k$. So, if $\mathcal{T}_k$ is the set of possible transformation matrices corresponding to the type of transformation at step $k$ (*e.g.*, image rotation), $\mathbf{T}_k \in \mathcal{T}_k$.

The generative model is shown in Fig. 1a and consists of picking a class $c$, drawing a vector of image pixel intensities $\mathbf{z}_0$ from a Gaussian, picking the first transformation matrix $\mathbf{T}_1$ from $\mathcal{T}_k$, applying this transformation to $\mathbf{z}_0$ and adding Gaussian noise to obtain $\mathbf{z}_1$, and repeating this process until the last transformation matrix $\mathbf{T}_K$ is drawn from $\mathcal{T}_K$ and is applied to $\mathbf{z}_{K-1}$ to obtain the observed data $\mathbf{z}_K$. The joint distribution is

$$p(c, \mathbf{z}_0, \mathbf{T}_1, \mathbf{z}_1, \dots, \mathbf{T}_K, \mathbf{z}_K) = p(c)p(\mathbf{z}_0|c) \prod_{k=1}^{K} p(\mathbf{T}_k)p(\mathbf{z}_k|\mathbf{z}_{k-1}, \mathbf{T}_k). \qquad (1)$$

The probability of class $c \in \{1, \dots, C\}$ is parameterized by $p(c) = \pi_c$ and the untransformed latent image has conditional density

$$p(\mathbf{z}_0|c) = \mathcal{N}(\mathbf{z}_0; \boldsymbol{\mu}_c, \boldsymbol{\Phi}_c), \qquad (2)$$

where $\mathcal{N}()$ is the normal distribution, $\boldsymbol{\mu}_c$ is the mean image for class $c$ and $\boldsymbol{\Phi}_c$ is the diagonal noise covariance matrix for class $c$. Notice that the noise modeled by $\boldsymbol{\Phi}_c$ gets transformed, so $\boldsymbol{\Phi}_c$ can model noise sources that depend on the transformations, such as background clutter and object deformations in images.

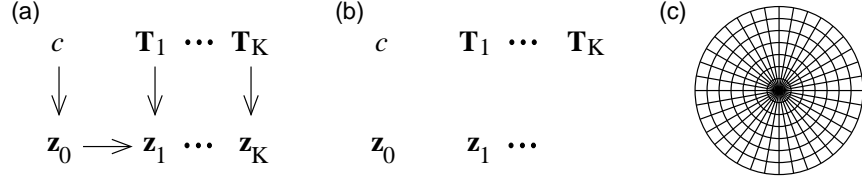

Figure 1: (a) The Bayesian network for a generative model that draws an image $\mathbf{z}_0$ from class $c$, applies a randomly drawn transformation matrix $\mathbf{T}_1$ of type 1 (*e.g.*, image rotation) to obtain $\mathbf{z}_1$, and so on, until a randomly drawn transformation matrix $\mathbf{T}_K$ of type $K$ (*e.g.*, image translation) is applied to obtain the observed image $\mathbf{z}_K$. (b) The Bayesian network for a factorized variational approximation to the posterior distribution, given $\mathbf{z}_K$. (c) When an image is measured on a discrete, radial 2-D grid, a scale and rotation correspond to a shift in the radial and angular coordinates.

The probability of transformation matrix $\mathbf{T}_k$ at step $k$ is $p(\mathbf{T}_k) = \lambda_{k,\mathbf{T}_k}$. (In our experiments, we often fix this to be uniform.) At each step, we assume a small amount of noise with diagonal covariance matrix $\boldsymbol{\Psi}$ is added to the image, so

$$p(\mathbf{z}_k|\mathbf{z}_{K-1}, \mathbf{T}_k) = \mathcal{N}(\mathbf{z}_k; \mathbf{T}_k\mathbf{z}_{k-1}, \boldsymbol{\Psi}). \tag{3}$$

$\mathbf{T}_k$ operates on $\mathbf{z}_{k-1}$ to produce a transformed image. In fact, $\mathbf{T}_k$ can be viewed as a permutation matrix that rearranges the pixels in $\mathbf{z}_{k-1}$. Usually, we assume $\boldsymbol{\Psi} = \psi\mathbf{I}$ and in our experiments we often set $\psi$ to a constant, small value, such as 0.01.

In (2001), an exact EM algorithm for learning this model is described. The sufficient statistics for $\pi_c$, $\boldsymbol{\mu}_c$ and $\boldsymbol{\Phi}_c$ are computed by averaging the derivatives of $\ln(\pi_c\mathcal{N}(\mathbf{z}_0; \boldsymbol{\mu}_c, \boldsymbol{\Phi}_c))$ over the posterior distribution,

$$p(c, \mathbf{z}_0|\mathbf{z}_K) = \sum_{\mathbf{T}_1}\cdots\sum_{\mathbf{T}_K} p(\mathbf{z}_0|c, \mathbf{T}_1, \dots, \mathbf{T}_K, \mathbf{z}_K)p(c, \mathbf{T}_1, \dots, \mathbf{T}_K|\mathbf{z}_K). \tag{4}$$

Since $\mathbf{z}_0, \dots, \mathbf{z}_K$ are jointly Gaussian given $c$ and $\mathbf{T}_1, \dots, \mathbf{T}_K$, $p(\mathbf{z}_0|c, \mathbf{T}_1, \dots, \mathbf{T}_K, \mathbf{z}_K)$ is Gaussian and its mean and covariance are computed using linear algebra. Also, $p(c, \mathbf{T}_1, \dots, \mathbf{T}_K|\mathbf{z}_K)$ is computed using linear algebra.

The problem with this direct approach is that the number of scalar operations in (4) is very large for large feature vectors and large sets of transformations. For $N \times N$ images, learning $C$ clusters under $N$ rotations, $N$ scales, $N$ $x$-translations and $N$ $y$-translations leads to $N^4$ terms in the summation. Since there are $N^2$ pixels, each term is computed using $N^2$ scalar operations. So, each iteration of EM takes $CN^6$ scalar operations per training case. For 10 classes and images of size $256 \times 256$, the direct approach takes $2.8 \times 10^{15}$ scalar operations per image for each iteration of EM.

We now describe how a variational technique for decoupling transformations, and a fast Fourier transform method for computing posterior probabilities can reduce the above number to $(C + 2\log N)N^2$ scalar operations. For 10 classes and images of size $256 \times 256$, the new method takes $2,752,512$ scalar operations per image for each iteration of EM.

## 3  Factorized variational technique

To simplify the computation of the required posterior in (4), we use a variational approximation (Jordan et al. 1998). As shown in Fig. 1b, our variational approximation is a completely factorized approximation to the true posterior:

$$p(c, \mathbf{z}_0, \mathbf{T}_1, \mathbf{z}_1, \dots, \mathbf{T}_K | \mathbf{z}_K) \approx q(c, \mathbf{z}_0, \mathbf{T}_1, \mathbf{z}_1, \dots, \mathbf{T}_K)$$

$$= q(c)q(\mathbf{z}_0)\Big(\prod_{k=1}^{K-1} q(\mathbf{T}_k)q(\mathbf{z}_k)\Big)q(\mathbf{T}_K). \tag{5}$$

The $q$-distributions are parameterized and these variational parameters are varied to make the approximation a good one. $p(c, \mathbf{z}_0 | \mathbf{z}_K) \approx q(c)q(\mathbf{z}_K)$, so the sufficient statistics can be readily determined from $q(c)$ and $q(\mathbf{z}_K)$. The variational parameters are $q(c) = \rho_c$, $q(\mathbf{T}_k) = \gamma_{k,\mathbf{T}_k}$, $q(\mathbf{z}_k) = \mathcal{N}(\mathbf{z}_k; \boldsymbol{\eta}_k, \boldsymbol{\Omega}_k)$.

The generalized EM algorithm (Neal and Hinton 1998) maximizes a lower bound on the log-likelihood of the observed image $\mathbf{z}_K$:

$$\mathcal{B} = \sum \int q(c, \mathbf{z}_0, \mathbf{T}_1, \mathbf{z}_1, \dots, \mathbf{T}_K) \ln \frac{p(c, \mathbf{z}_0, \mathbf{T}_1, \mathbf{z}_1, \dots, \mathbf{T}_K, \mathbf{z}_K)}{q(c, \mathbf{z}_0, \mathbf{T}_1, \mathbf{z}_1, \dots, \mathbf{T}_K)} \leq \ln p(\mathbf{z}_K). \tag{6}$$

In the E step, the variational parameters are adjusted to maximize $\mathcal{B}$ and in the M step, the model parameters are adjusted to maximize $\mathcal{B}$.

Assuming constant noise, $\boldsymbol{\Psi} = \psi\mathbf{I}$, the derivatives of $\mathcal{B}$ with respect to the variational parameters produce the following E-step updates:

$$\boldsymbol{\Omega}_0 \leftarrow \Big(\sum_c \rho_c \boldsymbol{\Phi}_c^{-1} + \psi^{-1}\mathbf{I}\Big)^{-1}$$

$$\boldsymbol{\eta}_0 \leftarrow \boldsymbol{\Omega}_0\Big(\sum_c \rho_c \boldsymbol{\Phi}_c^{-1}\boldsymbol{\mu}_c + \psi^{-1}\sum_{\mathbf{T}_1} \gamma_{1,\mathbf{T}_1}\mathbf{T}_1^{-1}\boldsymbol{\eta}_1\Big) \tag{7}$$

$$\rho_c \leftarrow \pi_c \exp\Big(-\frac{1}{2}\mathrm{tr}(\boldsymbol{\Omega}_0\boldsymbol{\Phi}_c^{-1}) - \frac{1}{2}(\boldsymbol{\eta}_0 - \boldsymbol{\mu}_c)'\boldsymbol{\Phi}_c^{-1}(\boldsymbol{\eta}_0 - \boldsymbol{\mu}_c)\Big)$$

$$\boldsymbol{\Omega}_k \leftarrow \frac{1}{2}\psi\mathbf{I}$$

$$\boldsymbol{\eta}_k \leftarrow \frac{1}{2}\Big(\sum_{\mathbf{T}_k} \gamma_{k,\mathbf{T}_k}\mathbf{T}_k\boldsymbol{\eta}_{k-1} + \sum_{\mathbf{T}_{k+1}} \gamma_{k+1,\mathbf{T}_{k+1}}\mathbf{T}_{k+1}^{-1}\boldsymbol{\eta}_{k+1}\Big) \tag{8}$$

$$\gamma_{k,\mathbf{T}_k} \leftarrow \lambda_{k,\mathbf{T}_k} \exp\Big(-\frac{1}{2}\mathrm{tr}(\boldsymbol{\Omega}_k\psi^{-1}) - \frac{1}{2}\psi^{-1}(\boldsymbol{\eta}_k - \mathbf{T}_k\boldsymbol{\eta}_{k-1})'(\boldsymbol{\eta}_k - \mathbf{T}_k\boldsymbol{\eta}_{k-1})\Big). \tag{9}$$

Each time the $\rho_c$'s are updated, they should be normalized and similarly for the $\gamma_{k,\mathbf{T}_k}$'s. One or more iterations of the above updates are applied for each training case and the variational parameters are stored for use in the M-step, and as the initial conditions for the next E-step.

The derivatives of $\mathcal{B}$ with respect to the model parameters produce the following M-step updates:

$$\pi_c \leftarrow \langle\rho_c\rangle$$

$$\boldsymbol{\mu}_c \leftarrow \langle\rho_c\boldsymbol{\eta}_0\rangle$$

$$\boldsymbol{\Phi}_c \leftarrow \langle\rho_c(\boldsymbol{\Omega}_0 + \mathrm{diag}((\boldsymbol{\eta}_0 - \boldsymbol{\mu}_c)(\boldsymbol{\eta}_0 - \boldsymbol{\mu}_c)'))\rangle, \tag{10}$$

where $\langle\rangle$ indicates an average over the training set.

This factorized variational inference technique is quite greedy, since at each step, the method approximates the posterior with one Gaussian. So, the method works best for a small number of steps (2 in our experiments).

## 4    Inference using fast Fourier transforms

The M-step updates described above take very few computations, but the E-step updates can be computationally burdensome. The dominant culprits are the computation of the distance of the form

$$d_{\mathbf{T}} = (\mathbf{g} - \mathbf{Th})'(\mathbf{g} - \mathbf{Th}) \tag{11}$$

in (9), for *all* possible transformations $\mathbf{T}$, and the computation of the form

$$\sum_{\mathbf{T}} \gamma_{\mathbf{T}} \mathbf{Th} \tag{12}$$

in (7) and (8).

Since the variational approximation is more accurate when the transformations are broken into fewer steps, it is a good idea to pack as many transformations into each step as possible. In our experiments, $x$-$y$ translations are applied in one step, and rotations are applied in another step. However, the number of possible $x$-$y$ translations in a $320 \times 240$ image is 76,800. So, 76,800 $d_{\mathbf{T}}$'s must be computed and the computation of each $d_{\mathbf{T}}$ uses a vector norm of size 76,800.

It turns out that if the data is defined on a coordinate system where the effect of a transformation is a *shift*, the above quantities can be computed very quickly using fast Fourier transforms (FFTs). For images measured on rectangular grids, an $x$-$y$ translation corresponds to a shift in the coordinate system. For images measured on a radial grid, such as the one shown in Fig. 1c, a scale and rotation corresponds to a shift in the coordinate system (Wolberg and Zokai 2000).

When updating the variational parameters, it is straightforward to convert them to the appropriate coordinate system, apply the FFT method and convert them back.

We now use a very different notation to describe the FFT method. The image is measured on a discrete grid and $\mathbf{x}$ is the $x$-$y$ coordinate of a pixel in the image ($\mathbf{x}$ is a 2-vector). The images $\mathbf{g}$ and $\mathbf{h}$ in (11) and (12) are written as functions of $\mathbf{x}$: $g(\mathbf{x})$, $h(\mathbf{x})$. In this representation, $\mathbf{T}$ is an integer 2-vector, corresponding to a shift in $\mathbf{x}$. So, (11) becomes

$$d(\mathbf{T}) = \sum_{\mathbf{x}} (g(\mathbf{x}) - h(\mathbf{x} + \mathbf{T}))^2 = \sum_{\mathbf{x}} (g(\mathbf{x})^2 - 2g(\mathbf{x})h(\mathbf{x} + \mathbf{T}) + h(\mathbf{x} + \mathbf{T})^2) \tag{13}$$

and (12) becomes

$$\sum_{\mathbf{T}} \gamma(\mathbf{T})h(\mathbf{x} + \mathbf{T}). \tag{14}$$

The common form is the correlation

$$f(\mathbf{T}) = \sum_{\mathbf{x}} g(\mathbf{x})h(\mathbf{x} + \mathbf{T}), \tag{15}$$

For an $N \times N$ grid, computing the correlation directly for all $\mathbf{T}$ takes $N^4$ scalar operations. The FFT can be used to compute the correlation in $N^2 \log N$ time. The FFTs $G(\boldsymbol{\omega})$ and $H(\boldsymbol{\omega})$ of $g$ and $h$ are computed in $N^2 \log N$ time. Then, the FFT $F(\boldsymbol{\omega})$ of $f$ is computed in $N^2$ as follows,

$$F(\boldsymbol{\omega}) = G(\boldsymbol{\omega})^* H(\boldsymbol{\omega}), \tag{16}$$

where "*" denotes complex conjugate. Then the inverse FFT $f(\mathbf{T})$ of $F(\boldsymbol{\omega})$ is computed in $N^2 \log N$ time.

Using this method, the posterior and sufficient statistics for all $N^2$ shifts in an $N \times N$ grid can be computed in $N^2 \log N$ time. Using this method along with the variational technique, $C$ classes, $N$ scales, $N$ rotations, $N$ $x$-translations and $N$ $y$-translations can be accounted for using $(C + 2 \log N)N^2$ scalar operations.

## 5 Results

In order to compare our new learning algorithm with the previously published result, we repeated the experiment on clustering head poses in 200 44x28 frames. We achieved essentially the same result, but in only 10 seconds as opposed to 40 minutes that the original algorithm needed to compete the task. Both algorithms were implemented in Matlab. It should be noted that the original algorithm tested only for 9 vertical and 9 horizontal shifts (81 combinations), while the new algorithm dealt with all 1232 possible discrete shifts. This makes the new algorithm 600 times faster on low resolution data. The speed-up is even more drastic at higher resolutions, and when rotations and scales are added, since the complexity of the original algorithm is $CN^6$, where $C$ is the number of classes and N is the number of pixels.

The speed-up promised in the abstract is based on our computations, but obviously we were not able to run the original algorithm on full 320x240 resolution data. To illustrate that the fast variational technique presented here can be efficiently used to learn data means in the presence of scale change, significant rotations and translations in the data, we captured 10 seconds of a video at 320x240 resolution and trained a two-stage transformation-invariant where the first stage modeled rotations and scales as shifts in the log-polar coordinate system and the second stage modeled all possible shifts as described above. In Fig. 2 we show the results of training an ordinary Gaussian model, shift-invariant model and finally the scale, rotation and shift invariant model on the sequence. We also show three frames from the sequence stabilized using the variational inference.

## 6 Conclusions

We describes how a tremendous speed-up in training transformation-invariant generative model can be achieved through the use of a variational technique for decoupling transformations, and a fast Fourier transform method for computing posterior probabilities. For $N \times N$ images, learning $C$ clusters under $N$ rotations, $N$ scales, $N$ $x$-translations and $N$ $y$-translations takes only $(C + 2 \log N)N^2$ scalar operations per iteration. In contrast, the original algorithm takes $CN^6$ operations to account for these transformations. In this way we were able to reduce the computation to only seconds per frame for the images of 320x240 resolution using a simple Matlab implementation.

This opens the door for generative models of pixel intensities in video to be efficiently used for transformation-invariant video summary and search. As opposed to most techniques used in computer vision today, the generative modeling approach provides the likelihood model useful for search or retrieval, automatic clustering of the data and the extensibility through adding new hidden variables.

The model described here could potentially be useful for other high-dimensional data, such as audio.

## References

Dempster, A. P., Laird, N. M., and Rubin, D. B. 1977. Maximum likelihood from incomplete data via the EM algorithm. *Proceedings of the Royal Statistical Society*, B-39:1–38.

Frey, B. J. and Jojic, N. 2001. Transformation invariant clustering and dimensionality reduction. *IEEE Transactions on Pattern Analysis and Machine Intelligence.* To appear. Available at `http://www.cs.utoronto.ca/∼frey`.

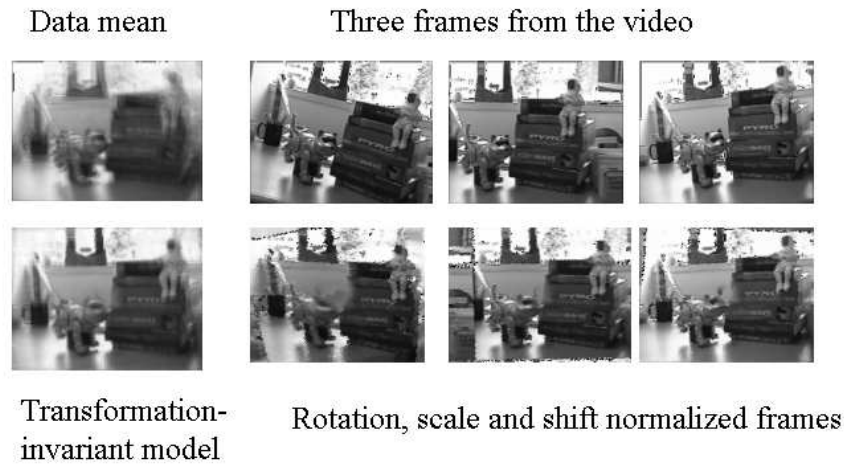

Figure 2: Learning a rotation, scale and translation invariant model on 320x240 video

Hinton, G. E., Dayan, P., and Revow, M. 1997. Modeling the manifolds of images of handwritten digits. *IEEE Transactions on Pattern Analysis and Machine Intelligence*, 8:65–74.

Jojic, N., Petrovic, N., Frey, B. J., and Huang, T. S. 2000. Transformed hidden markov models: Estimating mixture models of images and inferring spatial transformations in video sequences. In *Proceedings of the IEEE Conference on Computer Vision and Pattern Recognition*.

Jordan, M. I., Ghahramani, Z., Jaakkola, T. S., and Saul, L. K. 1998. An introduction to variational methods for graphical models. In Jordan, M. I., editor, *Learning in Graphical Models*. Kluwer Academic Publishers, Norwell MA.

LeCun, Y., Bottou, L., Bengio, Y., and Haffner, P. 1998. Gradient-based learning applied to document recognition. *Proceedings of the IEEE*, 86(11):2278–2324.

Neal, R. M. and Hinton, G. E. 1998. A view of the EM algorithm that justifies incremental, sparse, and other variants. In Jordan, M. I., editor, *Learning in Graphical Models*, pages 355–368. Kluwer Academic Publishers, Norwell MA.

Simard, P. Y., LeCun, Y., and Denker, J. 1993. Efficient pattern recognition using a new transformation distance. In Hanson, S. J., Cowan, J. D., and Giles, C. L., editors, *Advances in Neural Information Processing Systems 5*. Morgan Kaufmann, San Mateo CA.

Vasconcelos, N. and Lippman, A. 1998. Multiresolution tangent distance for affine-invariant classification. In Jordan, M. I., Kearns, M. I., and Solla, S. A., editors, *Advances in Neural Information Processing Systems 10*. MIT Press, Cambridge MA.

Wolberg, G. and Zokai, S. 2000. Robust image registration using log-polar transform. In *Proceedings IEEE Intl. Conference on Image Processing*, Vancouver, Canada.
